# Scaling laws and local minima in Hebbian ICA

**Magnus Rattray and Gleb Basalyga**
Department of Computer Science, University of Manchester,
Manchester M13 9PL, UK.
*magnus@cs.man.ac.uk, basalygg@cs.man.ac.uk*

## Abstract

We study the dynamics of a Hebbian ICA algorithm extracting a single non-Gaussian component from a high-dimensional Gaussian background. For both on-line and batch learning we find that a surprisingly large number of examples are required to avoid trapping in a sub-optimal state close to the initial conditions. To extract a skewed signal at least $O(N^2)$ examples are required for $N$-dimensional data and $O(N^3)$ examples are required to extract a symmetrical signal with non-zero kurtosis.

## 1  Introduction

Independent component analysis (ICA) is a statistical modelling technique which has attracted a significant amount of research interest in recent years (for a review, see Hyvärinen, 1999). The goal of ICA is to find a representation of data in terms of a combination of statistically independent variables. A number of neural learning algorithms have been applied to this problem, as detailed in the aforementioned review.

Theoretical studies of ICA algorithms have mainly focussed on asymptotic stability and efficiency, using the established results of stochastic approximation theory. However, in practice the transient stages of learning will often be more significant in determining the success of an algorithm. In this paper a Hebbian ICA algorithm is analysed in both on-line and batch mode, highlighting the critical importance of the transient dynamics. We find that a surprisingly large number of training examples are required in order to avoid trapping in a sub-optimal state close to the initial conditions. To detect a skewed signal at least $O(N^2)$ examples are required for $N$-dimensional data, while $O(N^3)$ examples are required for a symmetric signal with non-zero kurtosis. In addition, for on-line learning we show that the maximal initial learning rate which allows successful learning is unusually low, being $O(N^{-\frac{3}{2}})$ for a skewed signal and $O(N^{-2})$ for a symmetric signal.

In order to obtain a tractable model, we consider the limit of high-dimensional data and study an idealised data set in which a single non-Gaussian source is mixed into a large number of Gaussian sources. Recently, one of us considered a more general model in which an arbitrary, but relatively small, number of non-Gaussian sources were mixed into a high-dimensional Gaussian background (Rattray, 2002). In that work a solution to the dynamics of the on-line algorithm was obtained in closed form for $O(N)$ learning iterations and a simple solution to the asymptotic dynamics under the optimal learning rate decay was obtained. However, it was noted there that modelling the dynamics on an $O(N)$ timescale is not always appropriate, because the algorithm typically requires much longer in order to

escape from a class of metastable states close to the initial conditions. In order to elucidate this effect in greater detail we focus here on the simplest case of a single non-Gaussian source and we will limit our analysis to the dynamics close to the initial conditions.

In recent years a number of on-line learning algorithms, including back-propagation and Sanger's PCA algorithm, have been studied using techniques from statistical mechanics (see, for example, Biehl (1994); Biehl and Schwarze (1995); Saad and Solla (1995) and contributions in Saad (1998)). These analyses exploited the "self-averaging" property of certain macroscopic variables in order to obtain ordinary differential equations describing the deterministic evolution of these quantities over time in the large $N$ limit. In the present case the appropriate macroscopic quantity does not self-average and fluctuations have to be considered even in the limit. In this case it is more natural to model the on-line learning dynamics as a diffusion process (see, for example Gardiner, 1985).

## 2 Data Model

In order to apply the Hebbian ICA algorithm we must first sphere the data, ie. linearly transform the data so that it has zero mean and an identity covariance matrix. This can be achieved by standard transformations in a batch setting or for on-line learning an adaptive sphering algorithm, such as the one introduced by Cardoso and Laheld (1996), could be used. To simplify the analysis it is assumed here that the data has already been sphered. Without loss of generality it can also be assumed that the sources each have unit variance.

Each data point $x$ is generated from a noiseless linear mixture of sources which are decomposed into a single non-Gaussian source $s$ and $N-1$ uncorrelated Gaussian components, $n \sim \mathcal{N}(0, I_{N-1})$. We will also decompose the mixing matrix $A$ into a column vector $a_s$ and a $N \times (N-1)$ rectangular matrix $A_n$ associated with the non-Gaussian and Gaussian components respectively,

$$x = A \left[ \begin{array}{c} s \\ n \end{array} \right] = a_s s + A_n n . \tag{1}$$

We will consider both the on-line case, in which a new IID example $x^t$ is presented to the algorithm at each time $t$ and then discarded, and also the batch case, in which a finite set of examples are available to the algorithm. To conform with the model assumptions the mixing matrix $A$ must be unitary, which leads to the following constraints,

$$[a_s \; A_n] \left[ \begin{array}{c} a_s^{\mathrm{T}} \\ A_n^{\mathrm{T}} \end{array} \right] = a_s a_s^{\mathrm{T}} + A_n A_n^{\mathrm{T}} = I , \tag{2}$$

$$\left[ \begin{array}{c} a_s^{\mathrm{T}} \\ A_n^{\mathrm{T}} \end{array} \right] [a_s \; A_n] = \left[ \begin{array}{cc} a_s^{\mathrm{T}} a_s & a_s^{\mathrm{T}} A_n \\ A_n^{\mathrm{T}} a_s & A_n^{\mathrm{T}} A_n \end{array} \right] = \left[ \begin{array}{cc} 1 & 0 \\ 0 & I \end{array} \right] . \tag{3}$$

## 3 On-line learning

The goal of ICA is to find a vector $w$ such that the projection $y \equiv w^{\mathrm{T}} x \to \pm s$. Defining the overlap $R \equiv w^{\mathrm{T}} a_s$ we obtain,

$$\begin{aligned} y &= w^{\mathrm{T}}(a_s s + A_n n) \\ &= Rs + z \sqrt{\|w\|^2 - R^2} \quad \text{where} \quad z \sim \mathcal{N}(0, 1) , \end{aligned} \tag{4}$$

where we have made use of the constraint in eqn. (2). This assumes zero correlation between $w$ and $x$ which is true for on-line learning but is only strictly true for the first iteration of batch learning (see section 4). In the algorithm described below we impose a normalisation constraint on $w$ such that $\|w\| = 1$. In this case we see that the goal is to find $w$ such that $R \to \pm 1$.

A simple Hebbian (or anti-Hebbian) learning rule was studied by Hyvärinen and Oja (1998), who showed it to have a remarkably simple stability condition. We will consider the deflationary form in which a single source is learned at one time. The algorithm is closely related to Projection Pursuit algorithms, which seek interesting projections in high-dimensional data. A typical criteria for an interesting projection is to find one which is maximally non-Gaussian in some sense. Maximising some such measure (simple examples would be skewness or kurtosis) leads to the following simple algorithm (see Hyvärinen and Oja, 1998, for details). The change in $\boldsymbol{w}$ at time $t$ is given by,

$$\Delta \boldsymbol{w} = \eta\,\sigma\phi(y^t)\boldsymbol{x}^t\;;\;\text{ followed by normalisation such that }\;||\boldsymbol{w}|| = 1\;. \qquad (5)$$

Here, $\eta$ is the learning rate and $\phi(y)$ is some non-linear function which we will take to be at least three times differentiable. An even non-linearity, eg. $\phi(y) = y^2$, is appropriate for detecting asymmetric signals while a more common choice is an odd function, eg. $\phi(y) = y^3$ or $\phi(y) = \tanh(y)$, which can be used to detect symmetric non-Gaussian signals. In the latter case $\sigma \in \{-1, 1\}$ has to be chosen in order to ensure stability of the correct solution, as described by Hyvärinen and Oja (1998), either adaptively or using á priori knowledge. We set $\sigma = 1$ in the case of an even non-linearity. Remarkably, the same non-linearity can be used to separate both sub and super-Gaussian signals, in contrast to maximum likelihood methods for which this is typically not the case.

We can write the above algorithm as,

$$\boldsymbol{w}^{t+1} = \frac{\boldsymbol{w}^t + \eta\,\sigma\phi(y^t)\boldsymbol{x}^t}{\sqrt{1 + 2\eta\,\sigma\phi(y^t)y^t + \eta^2\phi^2(y^t)||\boldsymbol{x}^t||^2}}\;. \qquad (6)$$

For large $N$ and $\eta \leq O(N^{-1})$ (two different scalings will be considered below) we can expand out to get a weight decay normalisation,

$$\boldsymbol{w}^{t+1} \simeq \boldsymbol{w}^t + \eta\,\sigma\phi(y^t)\left(\boldsymbol{x}^t - y^t\boldsymbol{w}^t\right) - \tfrac{1}{2}\eta^2 N\phi^2(y^t)\boldsymbol{w}^t\;. \qquad (7)$$

Taking the dot-product with $\boldsymbol{a}_s$ gives the following update increment for the overlap $R$,

$$\Delta R = \eta\,\sigma\phi(y)\left(s^t - R^t y^t\right) - \tfrac{1}{2}\eta^2 N\phi^2(y^t)R^t\;, \qquad (8)$$

where we used the constraint in eqn. (3) to set $\boldsymbol{a}_s^{\mathrm{T}}\boldsymbol{x} = s$. Below we calculate the mean and variance of $\Delta R$ for two different scalings of the learning rate. Because the conditional distribution for $y$ given $s$ only depends on $R$ (setting $||\boldsymbol{w}|| = 1$ in eqn. 4) these expressions will depend only on $R$ and statistics of the non-Gaussian source distribution.

## 3.1 Dynamics close to the initial conditions

If the entries in $\boldsymbol{a}_s$ and $\boldsymbol{w}$ are initially of similar order then one would expect $R = O(N^{-\frac{1}{2}})$. This is the typical case if we consider a random and uncorrelated choice for $\boldsymbol{A}$ and the initial entries in $\boldsymbol{w}$. Larger initial values of $R$ could only be obtained with some prior knowledge of the mixing matrix which we will not assume. We will set $r \equiv R\sqrt{N}$ in the following discussion, where $r$ is assumed to be an $O(1)$ quantity. The discussion below is therefore restricted to describing the dynamics close to the initial conditions. For an account of the transient dynamics far from the initial conditions and the asymptotic dynamics close to an optimal solution, see Rattray (2002).

### 3.1.1 $\phi(y)$ even, $\kappa_3 \neq 0$

If the signal is asymmetrical then an even non-linearity can be used, for example $\phi(y) = y^2$ is a common choice. In this case the appropriate (ie. maximal) scaling for the learning rate is $O(N^{-\frac{3}{2}})$ and we set $\eta = \nu/N^{\frac{3}{2}}$ where $\nu$ is an $O(1)$ scaled learning rate parameter. In

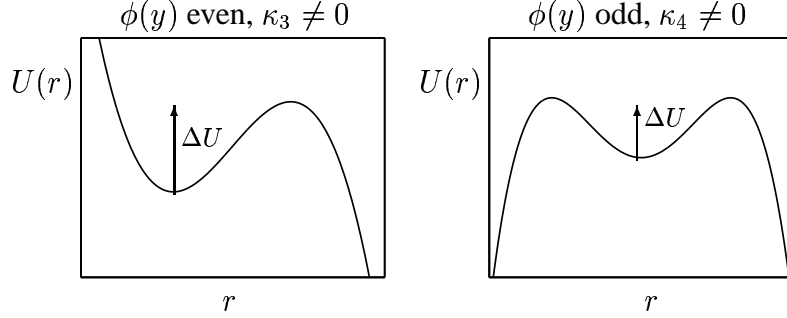

Figure 1: Close to the initial conditions (where $r \equiv \boldsymbol{w}^{\mathrm{T}} \boldsymbol{a}_s \sqrt{N} = O(1)$) the learning dynamics is equivalent to diffusion in a polynomial potential. For asymmetrical source distributions we can use an even non-linearity in which case the potential is cubic, as shown on the left. For symmetrical source distributions with non-zero kurtosis we should use an odd non-linearity in which case the potential is quartic, as shown on the right. The dynamics is initially confined in a metastable state near $r = 0$ with a potential barrier $\Delta U$.

this case we find that the mean and variance of the change in $r$ at each iteration are given by (to leading order in $N^{-1}$),

$$\mathrm{E}[\Delta r] \quad \simeq \quad \left(-\tfrac{1}{2}\langle\phi^2(z)\rangle\nu^2 r + \tfrac{1}{2}\kappa_3\langle\phi''(z)\rangle\nu\, r^2\right) N^{-2} \ , \tag{9}$$

$$\mathrm{Var}[\Delta r] \quad \simeq \quad \langle\phi^2(z)\rangle\nu^2 N^{-2} \ , \tag{10}$$

where $\kappa_3$ is the third cumulant of the source distribution (third central moment), which measures skewness, and brackets denote averages over $z \sim \mathcal{N}(0,1)$. We also find that $\mathrm{E}[(\Delta r)^n] = o(N^{-2})$ for integer $n > 2$. In this case the system can be described by a Fokker-Planck equation for large $N$ (see, for example, Gardiner, 1985) with a characteristic timescale of $O(N^2)$. The system is locally equivalent to a diffusion in the following cubic potential,

$$U(r) = \tfrac{1}{4}\langle\phi^2(z)\rangle\nu^2 r^2 - \tfrac{1}{6}\kappa_3\langle\phi''(z)\rangle\nu\, r^3 \ , \tag{11}$$

with a diffusion coefficient $D = \langle\phi^2(z)\rangle\nu^2$ which is independent of $r$. The shape of this potential is shown on the left of fig. 1. A potential barrier of $\Delta U$ must be overcome to escape a metastable state close to the initial conditions.

### 3.1.2 $\phi(y)$ odd, $\kappa_4 \neq 0$

If the signal is symmetrical, or only weakly asymmetrical, it will be necessary to use an odd non-linearity, for example $\phi(y) = y^3$ or $\phi(y) = \tanh(y)$ are popular choices. In this case a lower learning rate is required in order to achieve successful separation. The appropriate scaling for the learning rate is $O(N^{-2})$ and we set $\eta = \nu/N^2$ where again $\nu$ is an $O(1)$ scaled learning rate parameter. In this case we find that the mean and variance of the change in $r$ at each iteration are given by,

$$\mathrm{E}[\Delta r] \quad \simeq \quad \left(-\tfrac{1}{2}\langle\phi^2(z)\rangle\nu^2 r + \tfrac{1}{6}\kappa_4\langle\phi'''(z)\rangle\sigma\nu\, r^3\right) N^{-3} \ , \tag{12}$$

$$\mathrm{Var}[\Delta r] \quad \simeq \quad \langle\phi^2(z)\rangle\nu^2 N^{-3} \ , \tag{13}$$

where $\kappa_4$ is the fourth cumulant of the source distribution (measuring kurtosis) and brackets denote averages over $z \sim \mathcal{N}(0,1)$. Again the system can be described by a Fokker-Planck

equation for large $N$ but in this case the timescale for learning is $O(N^3)$, an order of $N$ slower than in the asymmetrical case. The system is locally equivalent to diffusion in the following quartic potential,

$$U(r) = \tfrac{1}{4}\langle \phi^2(z)\rangle \nu^2 r^2 - \tfrac{1}{24}|\kappa_4 \langle \phi'''(z)\rangle|\nu\, r^4 \; , \tag{14}$$

with a diffusion coefficient $D = \langle \phi^2(z)\rangle \nu^2$. We have assumed $\sigma = \text{Sign}(\kappa_4)$ which is a necessary condition for successful learning. In the case of a cubic non-linearity this is also the condition for stability of the optimal fixed point, although in general these two conditions may not be equivalent (Rattray, 2002). The shape of this potential is shown on the right of fig. 1 and again a potential barrier of $\Delta U$ must be overcome to escape a metastable state close to the initial conditions.

### 3.1.3 Escape times from a metastable state at $R = 0$

For large $\nu$ the dynamics of $r$ corresponds to an Ornstein-Uhlenbeck process with a Gaussian stationary distribution of fixed unit variance. Thus, if one chooses too large $\nu$ initially the dynamics will become localised close to $R = 0$ (recall, $R = r/\sqrt{N}$). As $\nu$ is reduced the potential barrier confining the dynamics is reduced. The timescale for escape for large $\nu$ (mean first passage time) is mainly determined by the effective size of the barrier (see, for example, Gardiner, 1985),

$$T_{\text{escape}} \;\propto\; (\delta t)^{-1} \exp\left(\frac{\Delta U}{D}\right)\;, \tag{15}$$

where $\Delta U$ is the potential barrier, $D$ is the diffusion coefficient and $\delta t$ is a unit of time in the diffusion process. For the two cases considered above we obtain,

$$T_{\text{escape}}^{\text{even}} \;\propto\; N^2 \exp\left[\frac{1}{12}\left(\frac{\langle \phi^2(z)\rangle \nu}{\kappa_3 \langle \phi''(z)\rangle}\right)^2\right] \quad \text{for even } \phi(y),\, \kappa_3 \neq 0\,, \quad [\nu \equiv \eta N^{\frac{3}{2}}]$$

$$T_{\text{escape}}^{\text{odd}} \;\propto\; N^3 \exp\left[\frac{3\langle \phi^2(z)\rangle \nu}{8\,|\kappa_4 \langle \phi'''(z)\rangle|}\right] \qquad \text{for odd } \phi(y),\, \kappa_4 \neq 0\,. \quad [\nu \equiv \eta N^2] \tag{16}$$

The constants of proportionality depend on the shape of the potential and not on $N$. As the learning rate parameter is reduced so the timescale for escape is also reduced. However, the choice of optimal learning rate is non-trivial and cannot be determined by considering only the leading order terms in $R$ as above, because although small $\nu$ will result in a quicker escape from the unstable fixed point near $R = 0$, this in turn will lead to a very slow learning transient after escape. Notice that escape time is shortest when the cumulants $\kappa_3$ or $\kappa_4$ are large, suggesting that deflationary ICA algorithms will tend to find these signals first.

From the above discussion one can draw two important conclusions. Firstly, the initial learning rate must be less than $O(N^{-1})$ initially in order to avoid trapping close to the initial conditions. Secondly, the number of iterations required to escape the initial transient will be greater than $O(N)$, resulting in an extremely slow initial stage of learning for large $N$. The most extreme case is for symmetric source distributions with non-zero kurtosis, in which case $O(N^3)$ learning iterations are required.

In fig. 2 we show results of learning with an asymmetric source (top) and uniform source (bottom) for different scaled learning rates. As the learning rate is increased (left to right) we observe that the dynamics becomes increasingly stochastic, with the potential barrier becoming increasingly significant (potential maxima are shown as dashed lines). For the largest value of learning rate ($\nu = 5$) the algorithm becomes trapped close to the initial conditions for the whole simulation time. From the time axis we observe that the learning timescale is $O(N^2)$ for the asymmetrical signal and $O(N^3)$ for the symmetric signal, as predicted by our theory.

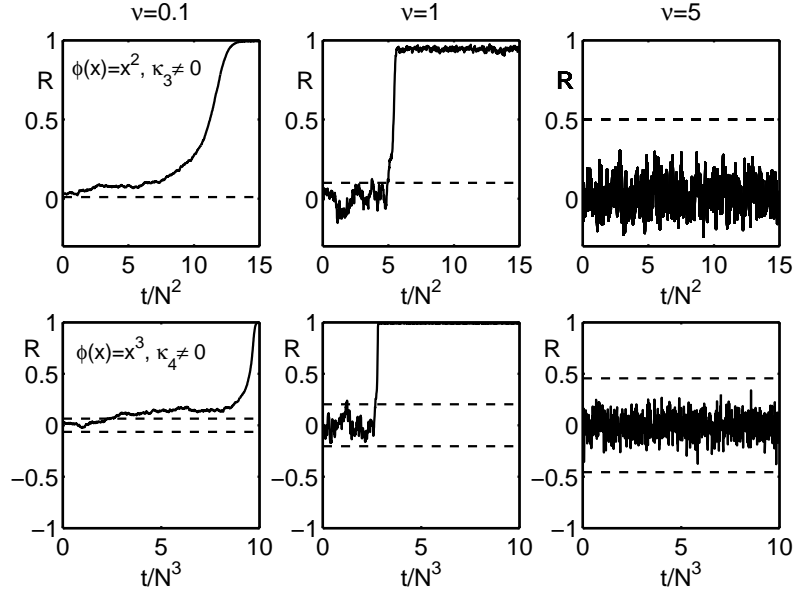

Figure 2: 100-dimensional data ($N = 100$) is produced from a mixture containing a single non-Gaussian source. In the top row we show results for a binary, asymmetrical source with skewness $\kappa_3 = 1.5$ and $\phi(x) = x^2$. In the bottom row we show results for a uniformly distributed source and $\phi(x) = x^3$. Each row shows learning with the same initial conditions and data but with different scaled learning rates (left to right $\nu = 0.1, 1$ and $5$) where $\nu \equiv \eta N^{\frac{3}{2}}$ (top) or $\nu \equiv \eta N^2$ (bottom). Dashed lines are maxima of the potentials in fig. 1.

## 4   Batch learning

The batch version of eqn. (5) for sufficiently small learning rates can be written,

$$\Delta \boldsymbol{w} \simeq \eta\,\sigma \sum_{t=1}^{P} \phi(y^t)\left(\boldsymbol{x}^t - y^t \boldsymbol{w}^t\right) \tag{17}$$

where $P$ is the number of training examples. Here we argue that such an update requires at least the same order of examples as in the on-line case, in order to be successful. Less data will result in a low signal-to-noise ratio initially and the possibility of trapping in a sub-optimal fixed point close to the initial conditions.

As in the on-line case we can write the update in terms of $R$,

$$\Delta R \simeq \eta\,\sigma \sum_{t=1}^{P} \phi(y^t)\left(s^t - y^t R^t\right)\ . \tag{18}$$

We make an assumption that successful learning is unlikely unless the initial increment in $R$ is in the desired direction. For example, with an asymmetric signal and quadratic non-linearity we require $\kappa_3 \Delta R > 0$ initially, while for a symmetric signal and odd non-linearity we require $R\,\Delta R > 0$. We have carried out simulations of batch learning which confirm that a relatively low percentage of runs in which the intial increment was incorrect result in successful learning compared to typical performance. As in the on-line case we observe that runs either succeed, in which case $R \to \pm 1$, or fail badly with $R$ remaining $O(N^{-\frac{1}{2}})$.

As before, $R = O(N^{-\frac{1}{2}})$ initially and we can therefore expand the right-hand side of eqn. (18) in orders of $R$ for large $N$. $\Delta R^{\mathrm{init}}$ ($\Delta R$ at the first iteration) is a sum over ran-

domly sampled terms, and the central limit theorem states that for large $P$ the distribution from which $\Delta R^{\text{init}}$ is sampled will be Gaussian, with mean and variance given by (to leading order in $R$),

$$\text{E}\left[\Delta R^{\text{init}}\right] \quad \simeq \quad \eta\,\sigma P\left[\tfrac{1}{2}\kappa_3\langle\phi''(z)\rangle R^2 + \tfrac{1}{6}\kappa_4\langle\phi'''(z)\rangle R^3\right]\;, \qquad (19)$$

$$\text{Var}\left[\Delta R^{\text{init}}\right] \quad \simeq \quad \eta^2 P\langle\phi^2(z)\rangle\;. \qquad (20)$$

Notice that the $\kappa_3$ term disappears in the case of an asymmetrical non-linearity, which is why we have left both terms in eqn. (19). The algorithm will be likely to fail when the standard deviation of $\Delta R^{\text{init}}$ is of the same order (or greater) than the mean. Since $R = O(N^{-\frac{1}{2}})$ initially, we see that this is true for $P = O(N^2)$ in the case of an even non-linearity and asymmetric signal, or for $P = O(N^3)$ in the case of an odd non-linearity and a signal with non-zero kurtosis. We expect these results to be necessary but not necessarily sufficient for successful learning, since we have only shown that this order of examples is the minimum required to avoid a low signal-to-noise ratio in the first learning iteration. A complete treatment of the batch learning problem would require much more sophisticated formulations such as the mean-field theory of Wong et al. (2000).

## 5 Conclusions and future work

In both the batch and on-line Hebbian ICA algorithm we find that a surprisingly large number of examples are required to avoid a sub-optimal fixed point close to the initial conditions. We expect simialr scaling laws to apply in the case of small numbers of non-Gaussian sources. Analysis of the square demixing problem appears to be much more challenging as in this case there may be no simple macroscopic description of the system for large $N$. It is therefore unclear at present whether ICA algorithms based on Maximum-likelihood and Information-theoretic principles (see, for example, Bell and Sejnowski, 1995; Amari et al., 1996; Cardoso and Laheld, 1996), which estimate a square demixing matrix, exhibit similar classes of fixed point to those studied here.

**Acknowledgements:** This work was supported by an EPSRC award (ref. GR/M48123). We would like to thank Jon Shapiro for useful comments on a preliminary version of this paper.

## References

S-I Amari, A Cichocki, and H H Yang. In D S Touretzky, M C Mozer, and M E Hasselmo, editors, *Neural Information Processing Systems 8*, pages 757–763. MIT Press, Cambridge MA, 1996.

A J Bell and T J Sejnowski. *Neural Computation*, 7:1129–1159, 1995.

M Biehl. *Europhys. Lett.*, 25:391–396, 1994.

M Biehl and H Schwarze. *J. Phys. A*, 28:643–656, 1995.

J-F Cardoso and B. Laheld. *IEEE Trans. on Signal Processing*, 44:3017–3030, 1996.

C. W. Gardiner. *Handbook of Stochastic Methods*. Springer-Verlag, New York, 1985.

A Hyvärinen. *Neural Computing Surveys*, 2:94–128, 1999.

A Hyvärinen and E Oja. *Signal Processing*, 64:301–313, 1998.

M Rattray. *Neural Computation*, 14, 2002 (in press).

D Saad, editor. *On-line Learning in Neural Networks*. Cambridge University Press, 1998.

D Saad and S A Solla. *Phys. Rev. Lett.*, 74:4337–4340, 1995.

K Y M Wong, S Li, and P Luo. In S A Solla, T K Leen, and K-R Müller, editors, *Neural Information Processing Systems 12*. MIT Press, Cambridge MA, 2000.